# TIME-SEQUENTIAL SELF-ORGANIZATION OF HIERARCHICAL NEURAL NETWORKS

Ronald H. Silverman
Cornell University Medical College, New York, NY 10021

Andrew S. Noetzel
Polytechnic University, Brooklyn, NY 11201

## ABSTRACT

Self-organization of multi-layered networks can be realized by time-sequential organization of successive neural layers. Lateral inhibition operating in the surround of firing cells in each layer provides for unsupervised capture of excitation patterns presented by the previous layer. By presenting patterns of increasing complexity, in co-ordination with network self-organization, higher levels of the hierarchy capture concepts implicit in the pattern set.

## INTRODUCTION

A fundamental difficulty in self-organization of hierarchical, multi-layered, networks of simple neuron-like cells is the determination of the direction of adjustment of synaptic link weights between neural layers not directly connected to input or output patterns. Several different approaches have been used to address this problem. One is to provide teaching inputs to the cells in internal layers of the hierarchy. Another is use of back-propagated error signals[1,2] from the uppermost neural layer, which is fixed to a desired output pattern. A third is the "competitive learning" mechanism,[3] in which a Hebbian synaptic modification rule is used, with mutual inhibition among cells of each layer preventing them from becoming conditioned to the same patterns.

The use of explicit teaching inputs is generally felt to be undesirable because such signals must, in essence, provide individual direction to each neuron in internal layers of the network. This requires extensive control signals, and is somewhat contrary to the notion of a self-organizing system.

Back-propagation provides direction for link weight modification of internal layers based on feedback from higher neural layers. This method allows true self-organization, but at the cost of specialized neural pathways over which these feedback signals must travel.

In this report, we describe a simple feed-forward method for self-organization of hierarchical neural networks. The method is a variation of the technique of competitive learning. It calls for successive neural layers to initiate modification of their afferent synaptic link weights only after the previous layer has completed its own self-organization. Additionally, the nature of the patterns captured can be controlled by providing an organized

group of pattern sets which would excite the lowermost (input) layer of the network in concert with training of successive layers. Such a collection of pattern sets might be viewed as a "lesson plan."

## MODEL

The network is composed of neuron-like cells, organized in hierarchical layers. Each cell is excited by variably weighted afferent connections from the outputs of the previous (lower) layer. Cells of the lowest layer take on the values of the input pattern. The cells themselves are of the McCulloch-Pitts type: they fire only after their excitation exceeds a threshold, and are otherwise inactive. Let $S_i(t) \, \varepsilon \{0,1\}$ be the state of cell i at time t. Let $w_{ij}$, a real number ranging from 0 to 1, be the weight, or strength, of the synapse connecting cell i to cell j. Let $e_{ij}$ be the local excitation of cell i at the synaptic connection from cell j. The excitation received along each synaptic connection is integrated locally over time as follows:

$$e_{ij}(t) = e_{ij}(t-1) + w_{ij}S_i(t) \qquad (1)$$

Synaptic connections may, therefore be viewed as capacitive. The total excitation, $E_j$, is the sum of the local excitations of cell j.

$$E_j(t) = \sum_i e_{ij}(t) \qquad (2)$$

The use of the time-integrated activity of a synaptic connection between two neurons, instead of the more usual instantaneous classification of neurons as "active" or "inactive", permits each synapse to provide a statistical measure of the activity of the input, which is assumed to be inherently stochastic. It also embodies the principle of learning based on locally available information and allows for implementations of the synapse as a capacitive element.

Over time, the total excitation of individual neurons on a give layer will increase. When excitation exceeds a threshold, $\theta$, then the neuron fires, otherwise it is inactive.

$$S_j(t) = 1 \text{ if } E_j(t) > \theta \qquad (3)$$
$$\text{else}$$
$$S_j(t) = 0$$

During a neuron's training phase, a modified Hebbian rule results in changes in afferent synaptic link weights such that, upon firing, synapses with integrated activity greater than mean activity are reinforced, and those with less than mean activity are weakened. More formally, if $S_j(t) = 1$ then the synapse weights are modified by

$$w_{ij}(t) = w_{ij}(t-1) + sign(e_{ij}(t) - \theta/n)k \cdot sine(\pi w_{ij}) \qquad (4)$$

Here, n represents the fan-in to a cell, and k is a small, positive constant. The "sign" function specifies the direction of change and the "sine" function determines the magnitude of change. The sine curve provides the property that intermediate

link weights are subject to larger modifications than weights near zero or saturation. This helps provide for stable end-states after learning.

Another effect of the integration of synaptic activity may be seen. A synapse of small weight is allowed to contribute to the firing of a cell (and hence have its weight incremented) if a series of patterns presented to the network consistently excite that synapse. The sequence of pattern presentations, therefore, becomes a factor in network self-organization.

Upon firing, the active cell inhibits other cells in its vicinity (lateral inhibition). This mechanism supports unsupervised, competitive learning. By preventing cells in the neighborhood of an active cell from modifying their afferent connections in response to a pattern, they are left available for capture of new patterns. Suppose there are n cells in a particular level. The lateral inhibitory mechanism is specified as follows:

$$\text{If} \quad S_j(t) = 1 \quad \text{then}$$
$$e_{ik}(t) = 0 \quad \text{for all i, for } k = (j-m)\bmod(n) \text{ to } (j+m)\bmod(n) \quad (5)$$

Here, m specifies the size of a "neighborhood." A neighborhood significantly larger than a pattern set will result in a number of untrained cells. A neighborhood smaller than the pattern set will tend to cause cells to attempt to capture more than one pattern.

Schematic representations of an individual cell and the network organization are provided in Figures 1 and 2.

It is the pattern generator, or "instructor", that controls the form that network organization will take. The initial set of patterns are repeated until the first layer is trained. Next, a new pattern set is used to excite the lowermost (trained) level of the network, and so, induce training in the next layer of the hierarchy. Each of the patterns of the new set is composed of elements (or subpatterns) of the old set. The structure of successive pattern sets is such that each set is either a more complex combination of elements from the previous set (as words are composed of letters) or a generalization of some concept implicit in the previous set (such as line orientation).

Network organization, as described above, requires some exchange of control signals between the network and the instructor. The instructor requires information regarding firing of cells during training in order to switch to a new patterns appropriately. Obviously, if patterns are switched before any cells fire, learning will either not take place or will be smeared over a number of patterns. If a single pattern excites the network until one or more cells are fully trained, subsequent presentation of a non-orthogonal pattern could cause the trained cell to fire before any naive cell because of its saturated link weights. The solution is simply to allow gradual training over the full complement of the pattern set. After a few firings, a new pattern should be provided. After a layer has been trained, the instructor provides a control signal to that layer which permanently fixes the layer's afferent synaptic link weights.

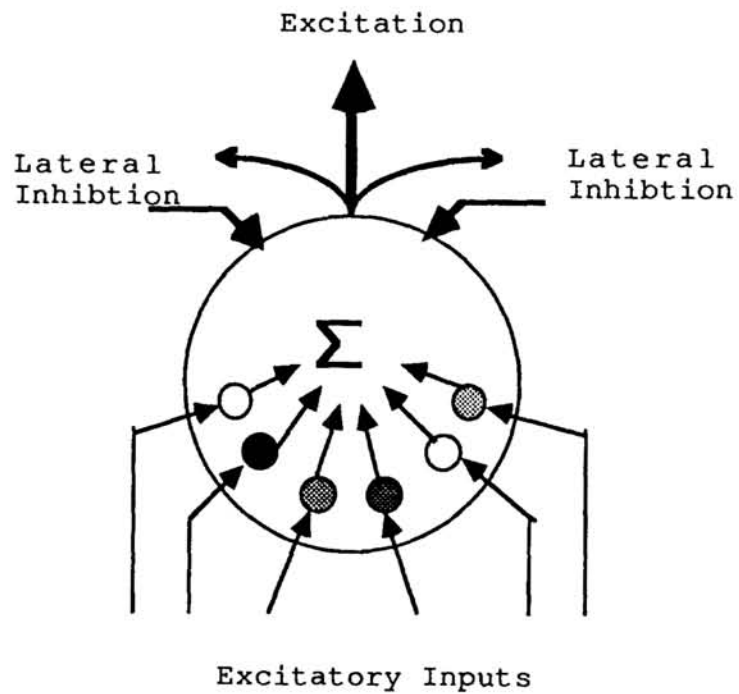

Excitation

Lateral
Inhibtion

Lateral
Inhibtion

$\Sigma$

Excitatory Inputs

Fig. 1. Schematic of neuron.
Shading of afferent synaptic connections
indicates variations in levels of local
time-integrated excitation.

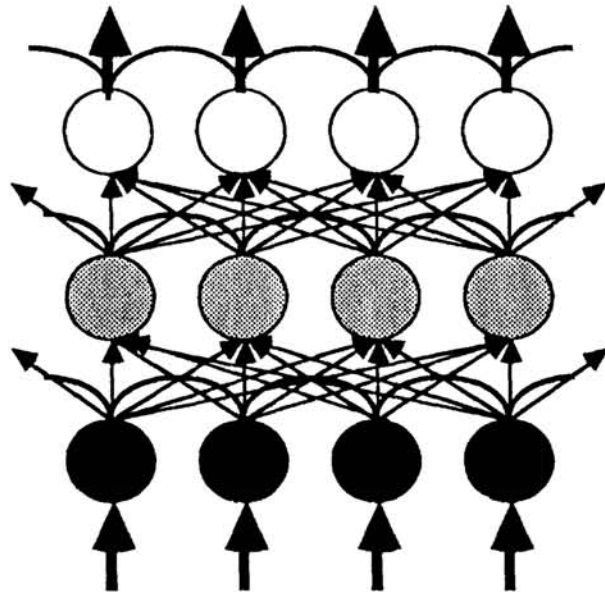

Fig. 2. Schematic of network showing
lateral inhibition and forward excitation.
Shading of neurons, indicating degree of
training, indicates time-sequential
organization of successive neural layers.

SIMULATIONS

As an example, simulations were run in which a network was taught to differentiate vertical from horizontal line orientation. This problem is of interest because it represents a case in which pattern sets cannot be separated by a single layer of connections. This is so because the set of vertical (or horizontal) lines has activity at all positions within the input matrix.

Two variations were simulated. In the first simulation, the input was a 4x4 matrix. This was completely connected with unidirectional links to 25 cells. These cells had fixed inhibitory connections to the nearest five cells on either side (using a circular arrangement), and excited, using complete connectivity, a ring of eight cells, with inhibition over the nearest neighbor on either side.

Initially, all excitatory link weights were small, random numbers. Each pattern of the initial input consisted of a single active row or column in the input matrix. Active elements had, during any clock cycle, a probability of 0.5 of being "on", while inactive elements had a 0.05 probability of being "on."

After exposure to the initial pattern set, all cells on the first layer captured some input pattern, and all eight patterns had been captured by two or more cells.

The next pattern set consisted of two subsets of four vertical and four horizontal lines. The individual lines were presented until a few firings took place within the trained layer, and then another line from the same subset was used to excite the network. After the upper layer responed with a few firings, and some training occured, the other set was used to excite the network in a similar manner. After five cycles, all cells on the uppermost layer had become sensitive, in a postionally independent manner, to lines of a vertical or a horizontal orientation. Due to lateral inhibition, adjacent cells developed opposite orientation specificities.

In the second simulation, a 6x6 input matrix was connected to six cells, which were, in turn, connected to two cells. For this network, the lateral inhibitory range extended over the entire set of cells of each layer. The initial input set consisted of six patterns, each of which was a pair of either vertical lines or horizontal lines. After excitation by this set, each of the six middle level cells became sensitized to one of the input patterns. Next, the set of vertical and horizontal patterns were grouped into two subsets: vertical lines and horizontal lines. Individual patterns from one subset were presented until a cell, of the previously trained layer, fired. After one of the two cells on the uppermost layer fired, the procedure was repeated with the pattern set of opposite orientation. After 25 cycles, the two cells on the uppermost layer had developed opposite orientation specificities. Each of these cells was shown to be responsive, in a positionally independent manner, to any single

line of appropriate orientation.

## CONCLUSION

Competitive learning mechanisms, when applied sequentially to successive layers in a hierarchical structure, can capture pattern elements, at lower levels of the hierarchy, and their generalizations, or abstractions, at higher levels.

In the above mechanism, learning is externally directed, not by explicit teaching signals or back-propagation, but by provision of instruction sets consisting of patterns of increasing complexity, to be input to the lowermost layer of the network in concert with successive organization of higher neural layers.

The central difficulty of this method involves the design of pattern sets - a procedure whose requirements may not be obvious in all cases. The method is, however, attractive due to its simplicity of concept and design, providing for multi-level self-organization without direction by elaborate control signals.

Several research goals suggest themselves: 1) simplification or elimination of control signals, 2) generalization of rules for structuring of pattern sets, 3) extension of this learning principle to recurrent networks, and 4) gaining a deeper understanding of the role of time as a factor in network self-organization.

## REFERENCES

1.  D. E. Rumelhart and G.E. Hinton, Nature 323, 533 (1986).
2.  K. A. Fukushima, Biol. Cybern. 55, 5 (1986).
3.  D. E. Rumelhart and D. Zipser, Cog. Sci. 9, 75 (1985).
